# Shallow vs. Deep Sum-Product Networks

**Olivier Delalleau**
Department of Computer Science and Operation Research
Université de Montréal
delallea@iro.umontreal.ca

**Yoshua Bengio**
Department of Computer Science and Operation Research
Université de Montréal
yoshua.bengio@umontreal.ca

## Abstract

We investigate the representational power of sum-product networks (computation networks analogous to neural networks, but whose individual units compute either products or weighted sums), through a theoretical analysis that compares deep (multiple hidden layers) vs. shallow (one hidden layer) architectures. We prove there exist families of functions that can be represented much more efficiently with a deep network than with a shallow one, i.e. with substantially fewer hidden units. Such results were not available until now, and contribute to motivate recent research involving learning of deep sum-product networks, and more generally motivate research in Deep Learning.

## 1 Introduction and prior work

Many learning algorithms are based on searching a family of functions so as to identify one member of said family which minimizes a training criterion. The choice of this family of functions and how members of that family are parameterized can be a crucial one. Although there is no universally optimal choice of parameterization or family of functions (or "architecture"), as demonstrated by the no-free-lunch results [37], it may be the case that some architectures are appropriate (or inappropriate) for a large class of learning tasks and data distributions, such as those related to Artificial Intelligence (AI) tasks [4]. Different families of functions have different characteristics that can be appropriate or not depending on the learning task of interest. One of the characteristics that has spurred much interest and research in recent years is **depth of the architecture**. In the case of a multi-layer neural network, depth corresponds to the number of (hidden and output) layers. A fixed-kernel Support Vector Machine is considered to have depth 2 [4] and boosted decision trees to have depth 3 [7]. Here we use the word *circuit* or *network* to talk about a directed acyclic graph, where each node is associated with some output value which can be computed based on the values associated with its predecessor nodes. The arguments of the learned function are set at the input nodes of the circuit (which have no predecessor) and the outputs of the function are read off the output nodes of the circuit. Different families of functions correspond to different circuits and allowed choices of computations in each node. Learning can be performed by changing the computation associated with a node, or rewiring the circuit (possibly changing the number of nodes). The depth of the circuit is the length of the longest path in the graph from an input node to an output node.

Deep Learning algorithms [3] are tailored to learning circuits with variable depth, typically greater than depth 2. They are based on the idea of *multiple levels of representation*, with the intuition that the raw input can be represented at different levels of abstraction, with more abstract features of the input or more abstract explanatory factors represented by deeper circuits. These algorithms are often based on unsupervised learning, opening the door to semi-supervised learning and efficient

use of large quantities of unlabeled data [3]. Analogies with the structure of the cerebral cortex (in particular the visual cortex) [31] and similarities between features learned with some Deep Learning algorithms and those hypothesized in the visual cortex [17] further motivate investigations into deep architectures. It has been suggested that deep architectures are more powerful in the sense of being able to more efficiently represent highly-varying functions [4, 3]. In this paper, we measure "efficiency" in terms of the number of computational units in the network. An efficient representation is important mainly because: (i) it uses less memory and is faster to compute, and (ii) given a fixed amount of training samples and computational power, better generalization is expected.

The first successful algorithms for training deep architectures appeared in 2006, with efficient training procedures for Deep Belief Networks [14] and deep auto-encoders [13, 27, 6], both exploiting the general idea of greedy layer-wise pre-training [6]. Since then, these ideas have been investigated further and applied in many settings, demonstrating state-of-the-art learning performance in object recognition [16, 28, 18, 15] and segmentation [20], audio classification [19, 10], natural language processing [9, 36, 21, 32], collaborative filtering [30], modeling textures [24], modeling motion [34, 33], information retrieval [29, 26], and semi-supervised learning [36, 22].

Poon and Domingos [25] introduced deep **sum-product networks** as a method to compute partition functions of tractable graphical models. These networks are analogous to traditional artificial neural networks but with nodes that compute either products or weighted sums of their inputs. Analogously to neural networks, we define "hidden" nodes as those nodes that are neither input nodes nor output nodes. If the nodes are organized in layers, we define the "hidden" layers to be those that are neither the input layer nor the output layer. Poon and Domingos [25] report experiments with networks much deeper (30+ hidden layers) than those typically used until now, e.g. in Deep Belief Networks [14, 3], where the number of hidden layers is usually on the order of three to five.

Whether such deep architectures have theoretical advantages compared to so-called "shallow" architectures (i.e. those with a single hidden layer) remains an open question. After all, in the case of a sum-product network, the output value can always be written as a sum of products of input variables (possibly raised to some power by allowing multiple connections from the same input), and consequently it is easily rewritten as a shallow network with a sum output unit and product hidden units. The argument supported by our theoretical analysis is that a deep architecture is able to compute some functions much more efficiently than a shallow one.

Until recently, very few theoretical results supported the idea that deep architectures could present an advantage in terms of representing some functions more efficiently. Most related results originate from the analysis of boolean circuits (see e.g. [2] for a review). Well-known results include the proof that solving the $n$-bit parity task with a depth-2 circuit requires an exponential number of gates [1, 38], and more generally that there exist functions computable with a polynomial-size depth-$k$ circuit that would require exponential size when restricted to depth $k - 1$ [11]. Another recent result on boolean circuits by Braverman [8] offers proof of a longstanding conjecture, showing that bounded-depth boolean circuits are unable to distinguish some (non-uniform) input distributions from the uniform distribution (i.e. they are "fooled" by such input distributions). In particular, Braverman's result suggests that shallow circuits can in general be fooled more easily than deep ones, i.e., that they would have more difficulty efficiently representing high-order dependencies (those involving many input variables).

It is not obvious that circuit complexity results (that typically consider only boolean or at least discrete nodes) are directly applicable in the context of typical machine learning algorithms such as neural networks (that compute continuous representations of their input). Orponen [23] surveys theoretical results in computational complexity that are relevant to learning algorithms. For instance, Håstad and Goldmann [12] extended some results to the case of networks of linear threshold units with positivity constraints on the weights. Bengio *et al.* [5, 7] investigate, respectively, complexity issues in networks of Gaussian radial basis functions and decision trees, showing intrinsic limitations of these architectures e.g. on tasks similar to the parity problem. Utgoff and Stracuzzi [35] informally discuss the advantages of depth in boolean circuit in the context of learning architectures. Bengio [3] suggests that some polynomials could be represented more efficiently by deep sum-product networks, but without providing any formal statement or proofs. This work partly addresses this void by demonstrating families of circuits for which a deep architecture can be exponentially more efficient than a shallow one in the context of real-valued polynomials.

Note that we do not address in this paper the problem of *learning* these parameters: even if an efficient deep representation exists for the function we seek to approximate, in general there is no

guarantee for standard optimization algorithms to easily converge to this representation. This paper focuses on the representational power of deep sum-product circuits compared to shallow ones, and studies it by considering particular families of target functions (to be represented by the learner).

We first formally define sum-product networks. We consider two families of functions represented by deep sum-product networks (families $\mathcal{F}$ and $\mathcal{G}$). For each family, we establish a lower bound on the minimal number of hidden units a depth-2 sum-product network would require to represent a function of this family, showing it is much less efficient than the deep representation.

## 2 Sum-product networks

**Definition 1.** *A sum-product network is a network composed of units that either compute the product of their inputs or a weighted sum of their inputs (where weights are strictly positive).*

Here, we restrict our definition of the generic term "sum-product network" to networks whose summation units have positive incoming weights[1], while others are called "negative-weight" networks.

**Definition 2.** *A "negative-weight" sum-product network may contain summation units whose weights are non-positive (i.e. less than or equal to zero).*

Finally, we formally define what we mean by *deep* vs. *shallow* networks in the rest of the paper.

**Definition 3.** *A "shallow" sum-product network contains a single hidden layer (i.e. a total of three layers when counting the input and output layers, and a depth equal to two).*

**Definition 4.** *A "deep" sum-product network contains more than one hidden layer (i.e. a total of at least four layers, and a depth at least three).*

## 3 The family $\mathcal{F}$

### 3.1 Definition

The first family of functions we study, denoted by $\mathcal{F}$, is made of functions built from deep sum-product networks that alternate layers of product and sum units with two inputs each (details are provided below). The basic idea we use here is that composing layers (i.e. using a deep architecture) is equivalent to using a factorized representation of the polynomial function computed by the network. Such a factorized representation can be exponentially more compact than its expansion as a sum of products (which can be associated to a shallow network with product units in its hidden layer and a sum unit as output). This is what we formally show in what follows.

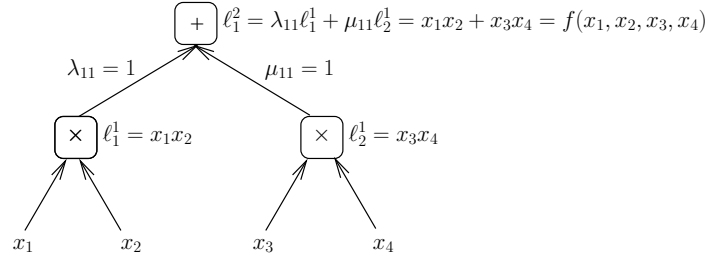

Figure 1: Sum-product network computing the function $f \in \mathcal{F}$ such that $i = \lambda_{11} = \mu_{11} = 1$.

Let $n = 4^i$, with $i$ a positive integer value. Denote by $\ell^0$ the input layer containing scalar variables $\{x_1, \ldots, x_n\}$, such that $\ell^0_j = x_j$ for $1 \leq j \leq n$. Now define $f \in F$ as any function computed by a sum-product network (deep for $i \geq 2$) composed of alternating product and sum layers:

- $\ell^{2k+1}_j = \ell^{2k}_{2j-1} \cdot \ell^{2k}_{2j}$ for $0 \leq k \leq i-1$ and $1 \leq j \leq 2^{2(i-k)-1}$
- $\ell^{2k}_j = \lambda_{jk} \ell^{2k-1}_{2j-1} + \mu_{jk} \ell^{2k-1}_{2j}$ for $1 \leq k \leq i$ and $1 \leq j \leq 2^{2(i-k)}$

where the weights $\lambda_{jk}$ and $\mu_{jk}$ of the summation units are strictly positive.

The output of the network is given by $f(x_1, \ldots, x_n) = \ell^{2i}_1 \in \mathbb{R}$, the unique unit in the last layer. The corresponding (shallow) network for $i = 1$ and additive weights set to one is shown in Figure 1

(this architecture is also the basic building block of bigger networks for $i > 1$). Note that both the input size $n = 4^i$ and the network's depth $2i$ increase with parameter $i$.

## 3.2 Theoretical results

The main result of this section is presented below in Corollary 1, providing a lower bound on the minimum number of hidden units required by a shallow sum-product network to represent a function $f \in \mathcal{F}$. The high-level proof sketch consists in the following steps:

(1) Count the number of unique products found in the polynomial representation of $f$ (Lemma 1 and Proposition 1).

(2) Show that the only possible architecture for a shallow sum-product network to compute $f$ is to have a hidden layer made of product units, with a sum unit as output (Lemmas 2 to 5).

(3) Conclude that the number of hidden units must be at least the number of unique products computed in step 3.2 (Lemma 6 and Corollary 1).

**Lemma 1.** *Any element $\ell_j^k$ can be written as a (positively) weighted sum of products of input variables, such that each input variable $x_t$ is used in exactly one unit of $\ell^k$. Moreover, the number $m_k$ of products found in the sum computed by $\ell_j^k$ does not depend on $j$ and obeys the following recurrence rule for $k \geq 0$: if $k + 1$ is odd, then $m_{k+1} = m_k^2$, otherwise $m_{k+1} = 2m_k$.*

*Proof.* We prove the lemma by induction on $k$. It is obviously true for $k = 0$ since $\ell_j^0 = x_j$. Assuming this is true for some $k \geq 0$, we consider two cases:

- If $k + 1$ is odd, then $\ell_j^{k+1} = \ell_{2j-1}^k \cdot \ell_{2j}^k$. By the inductive hypothesis, it is the product of two (positively) weighted sums of products of input variables, and no input variable can appear in both $\ell_{2j-1}^k$ and $\ell_{2j}^k$, so the result is also a (positively) weighted sum of products of input variables. Additionally, if the number of products in $\ell_{2j-1}^k$ and $\ell_{2j}^k$ is $m_k$, then $m_{k+1} = m_k^2$, since all products involved in the multiplication of the two units are different (since they use disjoint subsets of input variables), and the sums have positive weights.

  Finally, by the induction assumption, an input variable appears in exactly one unit of $\ell^k$. This unit is an input to a single unit of $\ell^{k+1}$, that will thus be the only unit of $\ell^{k+1}$ where this input variable appears.

- If $k + 1$ is even, then $\ell_j^{k+1} = \lambda_{jk}\ell_{2j-1}^k + \mu_{jk}\ell_{2j}^k$. Again, from the induction assumption, it must be a (positively) weighted sum of products of input variables, but with $m_{k+1} = 2m_k$ such products. As in the previous case, an input variable will appear in the single unit of $\ell^{k+1}$ that has as input the single unit of $\ell^k$ in which this variable must appear. $\square$

**Proposition 1.** *The number of products in the sum computed in the output unit $l_1^{2i}$ of a network computing a function in $\mathcal{F}$ is $m_{2i} = 2^{\sqrt{n}-1}$.*

*Proof.* We first prove by induction on $k \geq 1$ that for odd $k$, $m_k = 2^{2^{\frac{k+1}{2}}-2}$, and for even $k$, $m_k = 2^{2^{\frac{k}{2}}-1}$. This is obviously true for $k = 1$ since $2^{2^{\frac{1+1}{2}}-2} = 2^0 = 1$, and all units in $\ell^1$ are single products of the form $x_r x_s$. Assuming this is true for some $k \geq 1$, then:

- if $k + 1$ is odd, then from Lemma 1 and the induction assumption, we have:

$$m_{k+1} = m_k^2 = \left(2^{2^{\frac{k}{2}}-1}\right)^2 = 2^{2^{\frac{k}{2}+1}-2} = 2^{2^{\frac{(k+1)+1}{2}}-2}$$

- if $k + 1$ is even, then instead we have:

$$m_{k+1} = 2m_k = 2 \cdot 2^{2^{\frac{k+1}{2}}-2} = 2^{2^{\frac{(k+1)}{2}}-1}$$

which shows the desired result for $k + 1$, and thus concludes the induction proof. Applying this result with $k = 2i$ (which is even) yields

$$m_{2i} = 2^{2^{\frac{2i}{2}}-1} = 2^{\sqrt{2^{2i}}-1} = 2^{\sqrt{n}-1}.$$

$\square$

**Lemma 2.** *The products computed in the output unit $l_1^{2i}$ can be split in two groups, one with products containing only variables $x_1, \ldots, x_{\frac{n}{2}}$ and one containing only variables $x_{\frac{n}{2}+1}, \ldots, x_n$.*

*Proof.* This is obvious since the last unit is a "sum" unit that adds two terms whose inputs are these two groups of variables (see e.g. Fig. 1). □

**Lemma 3.** *The products computed in the output unit $l_1^{2i}$ involve more than one input variable.*

*Proof.* It is straightforward to show by induction on $k \geq 1$ that the products computed by $l_j^k$ all involve more than one input variable, thus it is true in particular for the output layer $(k = 2i)$. □

**Lemma 4.** *Any shallow sum-product network computing $f \in \mathcal{F}$ must have a "sum" unit as output.*

*Proof.* By contradiction, suppose the output unit of such a shallow sum-product network is multiplicative. This unit must have more than one input, because in the case that it has only one input, the output would be either a (weighted) sum of input variables (which would violate Lemma 3), or a single product of input variables (which would violate Proposition 1), depending on the type (sum or product) of the single input hidden unit. Thus the last unit must compute a product of two or more hidden units. It can be re-written as a product of two factors, where each factor corresponds to either one hidden unit, or a product of multiple hidden units (it does not matter here which specific factorization is chosen among all possible ones). Regardless of the type (sum or product) of the hidden units involved, those two factors can thus be written as weighted sums of products of variables $x_t$ (with positive weights, and input variables potentially raised to powers above one). From Lemma 1, both $x_1$ and $x_n$ must be present in the final output, and thus they must appear in at least one of these two factors. Without loss of generality, assume $x_1$ appears in the first factor. Variables $x_{\frac{n}{2}+1}, \ldots, x_n$ then cannot be present in the second factor, since otherwise one product in the output would contain both $x_1$ and one of these variables (this product cannot cancel out since weights must be positive), violating Lemma 2. But with a similar reasoning, since as a result $x_n$ must appear in the first factor, variables $x_1, \ldots, x_{\frac{n}{2}}$ cannot be present in the second factor either. Consequently, no input variable can be present in the second factor, leading to the desired contradiction. □

**Lemma 5.** *Any shallow sum-product network computing $f \in \mathcal{F}$ must have only multiplicative units in its hidden layer.*

*Proof.* By contradiction, suppose there exists a "sum" unit in the hidden layer, written $s = \sum_{t \in S} \alpha_t x_t$ with $S$ the set of input indices appearing in this sum, and $\alpha_t > 0$ for all $t \in S$. Since according to Lemma 4 the output unit must also be a sum (and have positive weights according to Definition 1), then the final output will also contain terms of the form $\beta_t x_t$ for $t \in S$, with $\beta_t > 0$. This violates Lemma 3, establishing the contradiction. □

**Lemma 6.** *Any shallow negative-weight sum-product network (see Definition 2) computing $f \in \mathcal{F}$ must have at least $2^{\sqrt{n}-1}$ hidden units, if its output unit is a sum and its hidden units are products.*

*Proof.* Such a network computes a weighted sum of its hidden units, where each hidden unit is a product of input variables, i.e. its output can be written as $\Sigma_j w_j \Pi_t x_t^{\gamma_{jt}}$ with $w_j \in \mathbb{R}$ and $\gamma_{jt} \in \{0,1\}$. In order to compute a function in $\mathcal{F}$, this shallow network thus needs a number of hidden units at least equal to the number of unique products in that function. From Proposition 1, this number is equal to $2^{\sqrt{n}-1}$. □

**Corollary 1.** *Any shallow sum-product network computing $f \in \mathcal{F}$ must have at least $2^{\sqrt{n}-1}$ hidden units.*

*Proof.* This is a direct corollary of Lemmas 4 (showing the output unit is a sum), 5 (showing that hidden units are products), and 6 (showing the desired result for any shallow network with this specific structure – regardless of the sign of weights). □

### 3.3 Discussion

Corollary 1 above shows that in order to compute some function in $\mathcal{F}$ with $n$ inputs, the number of units in a shallow network has to be at least $2^{\sqrt{n}-1}$, (i.e. grows exponentially in $\sqrt{n}$). On another hand, the total number of units in the deep (for $i > 1$) network computing the same function, as described in Section 3.1, is equal to $1 + 2 + 4 + 8 + \ldots + 2^{2i-1}$ (since all units are binary), which is also equal to $2^{2i} - 1 = n - 1$ (i.e. grows only quadratically in $\sqrt{n}$). **It shows that some deep sum-product network with n inputs and depth $\mathbf{O}(\log \mathbf{n})$ can represent with $\mathbf{O}(\mathbf{n})$ units what would require $\mathbf{O}(2^{\sqrt{\mathbf{n}}})$ units for a depth-2 network.** Lemma 6 also shows a similar result regardless of the sign of the weights in the summation units of the depth-2 network, but assumes a specific architecture for this network (products in the hidden layer with a sum as output).

## 4 The family $\mathcal{G}$

In this section we present similar results with a different family of functions, denoted by $\mathcal{G}$. Compared to $\mathcal{F}$, one important difference of deep sum-product networks built to define functions in $\mathcal{G}$ is that they can vary their input size independently of their depth. Their analysis thus provides additional insight when comparing the representational efficiency of deep vs. shallow sum-product networks in the case of a fixed dataset.

### 4.1 Definition

Networks in family $\mathcal{G}$ also alternate sum and product layers, but their units have as inputs all units from the previous layer *except one*. More formally, define the family $\mathcal{G} = \cup_{n \geq 2, i \geq 0} \mathcal{G}_{in}$ of functions represented by sum-product networks, where the sub-family $\mathcal{G}_{in}$ is made of all sum-product networks with $n$ input variables and $2i + 2$ layers (including the input layer $\ell^0$), such that:

1. $\ell^1$ contains summation units; further layers alternate multiplicative and summation units.
2. Summation units have positive weights.
3. All layers are of size $n$, except the last layer $\ell^{2i+1}$ that contains a single sum unit that sums all units in the previous layer $\ell^{2i}$.
4. In each layer $\ell^k$ for $1 \leq k \leq 2i$, each unit $\ell^k_j$ takes as inputs $\{\ell^{k-1}_m | m \neq j\}$.

An example of a network belonging to $\mathcal{G}_{1,3}$ (i.e. with three layers and three input variables) is shown in Figure 2.

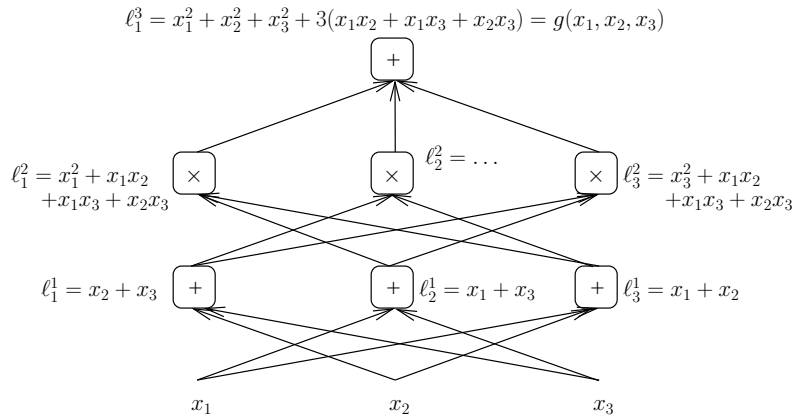

$$\ell^3_1 = x_1^2 + x_2^2 + x_3^2 + 3(x_1 x_2 + x_1 x_3 + x_2 x_3) = g(x_1, x_2, x_3)$$

$\ell^2_1 = x_1^2 + x_1 x_2 + x_1 x_3 + x_2 x_3$
$\ell^2_2 = \ldots$
$\ell^2_3 = x_3^2 + x_1 x_2 + x_1 x_3 + x_2 x_3$

$\ell^1_1 = x_2 + x_3$
$\ell^1_2 = x_1 + x_3$
$\ell^1_3 = x_1 + x_2$

$x_1 \quad x_2 \quad x_3$

Figure 2: Sum-product network computing a function of $\mathcal{G}_{1,3}$ (summation units' weights are all 1's).

### 4.2 Theoretical results

The main result is stated in Proposition 3 below, establishing a lower bound on the number of hidden units of a shallow sum-product network computing $g \in \mathcal{G}$. The proof sketch is as follows:

1. We show that the polynomial expansion of $g$ must contain a large set of products (Proposition 2 and Corollary 2).
2. We use both the number of products in that set as well as their degree to establish the desired lower bound (Proposition 3).

We will also need the following lemma, which states that when $n-1$ items each belong to $n-1$ sets among a total of $n$ sets, then we can associate to each item one of the sets it belongs to without using the same set for different items.

**Lemma 7.** *Let $S_1, \ldots, S_n$ be $n$ sets ($n \geq 2$) containing elements of $\{P_1, \ldots, P_{n-1}\}$, such that for any $q, r$, $|\{r|P_q \in S_r\}| \geq n-1$ (i.e. each element $P_q$ belongs to at least $n-1$ sets). Then there exist $r_1, \ldots, r_{n-1}$ different indices such that $P_q \in S_{r_q}$ for $1 \leq q \leq n-1$.*

*Proof.* Omitted due to lack of space (very easy to prove by construction). □

**Proposition 2.** *For any $0 \leq j \leq i$, and any product of variables $P = \Pi_{t=1}^n x_t^{\alpha_t}$ such that $\alpha_t \in \mathbb{N}$ and $\sum_t \alpha_t = (n-1)^j$, there exists a unit in $\ell^{2j}$ whose computed value, when expanded as a weighted sum of products, contains $P$ among these products.*

*Proof.* We prove this proposition by induction on $j$.

First, for $j = 0$, this is obvious since any $P$ of this form must be made of a single input variable $x_t$, that appears in $\ell_t^0 = x_t$.

Suppose now the proposition is true for some $j < i$. Consider a product $P = \Pi_{t=1}^n x_t^{\alpha_t}$ such that $\alpha_t \in \mathbb{N}$ and $\sum_t \alpha_t = (n-1)^{j+1}$. $P$ can be factored in $n-1$ sub-products of degree $(n-1)^j$, i.e. written $P = P_1 \ldots P_{n-1}$ with $P_q = \Pi_{t=1}^n x_t^{\beta_{qt}}$, $\beta_{qt} \in \mathbb{N}$ and $\sum_t \beta_{qt} = (n-1)^j$ for all $q$. By the induction hypothesis, each $P_q$ can be found in at least one unit $\ell_{k_q}^{2j}$. As a result, by property 4 (in the definition of family $\mathcal{G}$), each $P_q$ will also appear in the additive layer $\ell^{2j+1}$, in at least $n-1$ different units (the only sum unit that may not contain $P_q$ is the one that does not have $\ell_{k_q}^{2j}$ as input).

By Lemma 7, we can thus find a set of units $\ell_{r_q}^{2j+1}$ such that for any $1 \leq q \leq n-1$, the product $P_q$ appears in $\ell_{r_q}^{2j+1}$, with indices $r_q$ being different from each other. Let $1 \leq s \leq n$ be such that $s \neq r_q$ for all $q$. Then, from property 4 of family $\mathcal{G}$, the multiplicative unit $\ell_s^{2(j+1)}$ computes the product $\Pi_{q=1}^{n-1} \ell_{r_q}^{2j+1}$, and as a result, when expanded as a sum of products, it contains in particular $P_1 \ldots P_{n-1} = P$. The proposition is thus true for $j+1$, and by induction, is true for all $j \leq i$. □

**Corollary 2.** *The output $g_{in}$ of a sum-product network in $\mathcal{G}_{in}$, when expanded as a sum of products, contains all products of variables of the form $\Pi_{t=1}^n x_t^{\alpha_t}$ such that $\alpha_t \in \mathbb{N}$ and $\sum_t \alpha_t = (n-1)^i$.*

*Proof.* Applying Proposition 2 with $j = i$, we obtain that all products of this form can be found in the multiplicative units of $\ell^{2i}$. Since the output unit $\ell_1^{2i+1}$ computes a sum of these multiplicative units (weighted with positive weights), those products are also present in the output. □

**Proposition 3.** *A shallow negative-weight sum-product network computing $g_{in} \in \mathcal{G}_{in}$ must have at least $(n-1)^i$ hidden units.*

*Proof.* First suppose the output unit of the shallow network is a sum. Then it may be able to compute $g_{in}$, assuming we allow multiplicative units in the hidden layer in the hidden layer to use powers of their inputs in the product they compute (which we allow here for the proof to be more generic). However, it will require at least as many of these units as the number of unique products that can be found in the expansion of $g_{in}$. In particular, from Corollary 2, it will require at least the number of unique tuples of the form $(\alpha_1, \ldots, \alpha_n)$ such that $\alpha_t \in \mathbb{N}$ and $\sum_{t=1}^n \alpha_t = (n-1)^i$. Denoting $d_{ni} = (n-1)^i$, this number is known to be equal to $\binom{n+d_{ni}-1}{d_{ni}}$, and it is easy to verify it is higher than (or equal to) $d_{ni}$ for any $n \geq 2$ and $i \geq 0$.

Now suppose the output unit is multiplicative. Then there can be no multiplicative hidden unit, otherwise it would mean one could factor some input variable $x_t$ in the computed function output: this is not possible since by Corollary 2, for any variable $x_t$ there exist products in the output function that do not involve $x_t$. So all hidden units must be additive, and since the computed function contains products of degree $d_{ni}$, there must be at least $d_{ni}$ such hidden units. □

### 4.3 Discussion

Proposition 3 shows that in order to compute the same function as $g_{in} \in \mathcal{G}_{in}$, the number of units in the shallow network has to grow exponentially in $i$, i.e. in the network's depth (while the deep network's size grows linearly in $i$). The shallow network also needs to grow polynomially in the number of input variables $n$ (with a degree equal to $i$), while the deep network grows only linearly in $n$. **It means that some deep sum-product network with $n$ inputs and depth $O(i)$ can represent with $O(ni)$ units what would require $O((n-1)^i)$ units for a depth-2 network**.

Note that in the similar results found for family $\mathcal{F}$, the depth-2 network computing the same function as a function in $\mathcal{F}$ had to be constrained to either have a specific combination of sum and hidden units (in Lemma 6) or to have non-negative weights (in Corollary 1). On the contrary, the result presented here for family $\mathcal{G}$ holds without requiring any of these assumptions.

## 5 Conclusion

We compared a deep sum-product network and a shallow sum-product network representing the same function, taken from two families of functions $\mathcal{F}$ and $\mathcal{G}$. For both families, we have shown that the number of units in the shallow network has to grow exponentially, compared to a linear growth in the deep network, so as to represent the same functions. The deep version thus offers a much more compact representation of the same functions.

This work focuses on two specific families of functions: finding more general parameterization of functions leading to similar results would be an interesting topic for future research. Another open question is whether it is possible to represent such functions only approximately (e.g. up to an error bound $\epsilon$) with a much smaller shallow network. Results by Braverman [8] on boolean circuits suggest that similar results as those presented in this paper may still hold, but this topic has yet to be formally investigated in the context of sum-product networks. A related problem is also to look into functions defined only on discrete input variables: our proofs do not trivially extend to this situation because we cannot assume anymore that two polynomials yielding the same output values must have the same expansion coefficients (since the number of input combinations becomes finite).

**Acknowledgments**

The authors would like to thank Razvan Pascanu and David Warde-Farley for their help in improving this manuscript, as well as the anonymous reviewers for their careful reviews. This work was partially funded by NSERC, CIFAR, and the Canada Research Chairs.

## Footnotes

[1]This condition is required by some of the proofs presented here.

## References

[1] Ajtai, M. (1983). $\sum_1^1$-formulae on finite structures. *Annals of Pure and Applied Logic*, **24**(1), 1–48.

[2] Allender, E. (1996). Circuit complexity before the dawn of the new millennium. In *16th Annual Conference on Foundations of Software Technology and Theoretical Computer Science*, pages 1–18. Lecture Notes in Computer Science 1180, Springer Verlag.

[3] Bengio, Y. (2009). Learning deep architectures for AI. *Foundations and Trends in Machine Learning*, **2**(1), 1–127. Also published as a book. Now Publishers, 2009.

[4] Bengio, Y. and LeCun, Y. (2007). Scaling learning algorithms towards AI. In L. Bottou, O. Chapelle, D. DeCoste, and J. Weston, editors, *Large Scale Kernel Machines*. MIT Press.

[5] Bengio, Y., Delalleau, O., and Le Roux, N. (2006). The curse of highly variable functions for local kernel machines. In *NIPS'05*, pages 107–114. MIT Press, Cambridge, MA.

[6] Bengio, Y., Lamblin, P., Popovici, D., and Larochelle, H. (2007). Greedy layer-wise training of deep networks. In *NIPS 19*, pages 153–160. MIT Press.

[7] Bengio, Y., Delalleau, O., and Simard, C. (2010). Decision trees do not generalize to new variations. *Computational Intelligence*, **26**(4), 449–467.

[8] Braverman, M. (2011). Poly-logarithmic independence fools bounded-depth boolean circuits. *Communications of the ACM*, **54**(4), 108–115.

[9] Collobert, R. and Weston, J. (2008). A unified architecture for natural language processing: Deep neural networks with multitask learning. In *ICML 2008*, pages 160–167.

[10] Dahl, G. E., Ranzato, M., Mohamed, A., and Hinton, G. E. (2010). Phone recognition with the mean-covariance restricted boltzmann machine. In *Advances in Neural Information Processing Systems (NIPS)*.

[11] Håstad, J. (1986). Almost optimal lower bounds for small depth circuits. In *Proceedings of the 18th annual ACM Symposium on Theory of Computing*, pages 6–20, Berkeley, California. ACM Press.

[12] Håstad, J. and Goldmann, M. (1991). On the power of small-depth threshold circuits. *Computational Complexity*, **1**, 113–129.

[13] Hinton, G. E. and Salakhutdinov, R. (2006). Reducing the dimensionality of data with neural networks. *Science*, **313**(5786), 504–507.

[14] Hinton, G. E., Osindero, S., and Teh, Y. (2006). A fast learning algorithm for deep belief nets. *Neural Computation*, **18**, 1527–1554.

[15] Kavukcuoglu, K., Sermanet, P., Boureau, Y.-L., Gregor, K., Mathieu, M., and LeCun, Y. (2010). Learning convolutional feature hierarchies for visual recognition. In *NIPS'10*.

[16] Larochelle, H., Erhan, D., Courville, A., Bergstra, J., and Bengio, Y. (2007). An empirical evaluation of deep architectures on problems with many factors of variation. In *ICML'07*, pages 473–480. ACM.

[17] Lee, H., Ekanadham, C., and Ng, A. (2008). Sparse deep belief net model for visual area V2. In *NIPS'07*, pages 873–880. MIT Press, Cambridge, MA.

[18] Lee, H., Grosse, R., Ranganath, R., and Ng, A. Y. (2009a). Convolutional deep belief networks for scalable unsupervised learning of hierarchical representations. In *ICML 2009*. Montreal (Qc), Canada.

[19] Lee, H., Pham, P., Largman, Y., and Ng, A. (2009b). Unsupervised feature learning for audio classification using convolutional deep belief networks. In *NIPS'09*, pages 1096–1104.

[20] Levner, I. (2008). *Data Driven Object Segmentation*. Ph.D. thesis, Department of Computer Science, University of Alberta.

[21] Mnih, A. and Hinton, G. E. (2009). A scalable hierarchical distributed language model. In *NIPS'08*, pages 1081–1088.

[22] Mobahi, H., Collobert, R., and Weston, J. (2009). Deep learning from temporal coherence in video. In *ICML'2009*, pages 737–744.

[23] Orponen, P. (1994). Computational complexity of neural networks: a survey. *Nordic Journal of Computing*, **1**(1), 94–110.

[24] Osindero, S. and Hinton, G. E. (2008). Modeling image patches with a directed hierarchy of markov random field. In *NIPS'07*, pages 1121–1128, Cambridge, MA. MIT Press.

[25] Poon, H. and Domingos, P. (2011). Sum-product networks: A new deep architecture. In *UAI'2011*, Barcelona, Spain.

[26] Ranzato, M. and Szummer, M. (2008). Semi-supervised learning of compact document representations with deep networks. In *ICML*.

[27] Ranzato, M., Poultney, C., Chopra, S., and LeCun, Y. (2007). Efficient learning of sparse representations with an energy-based model. In *NIPS'06*, pages 1137–1144. MIT Press.

[28] Ranzato, M., Boureau, Y.-L., and LeCun, Y. (2008). Sparse feature learning for deep belief networks. In *NIPS'07*, pages 1185–1192, Cambridge, MA. MIT Press.

[29] Salakhutdinov, R. and Hinton, G. E. (2007). Semantic hashing. In *Proceedings of the 2007 Workshop on Information Retrieval and applications of Graphical Models (SIGIR 2007)*, Amsterdam. Elsevier.

[30] Salakhutdinov, R., Mnih, A., and Hinton, G. E. (2007). Restricted Boltzmann machines for collaborative filtering. In *ICML 2007*, pages 791–798, New York, NY, USA.

[31] Serre, T., Kreiman, G., Kouh, M., Cadieu, C., Knoblich, U., and Poggio, T. (2007). A quantitative theory of immediate visual recognition. *Progress in Brain Research, Computational Neuroscience: Theoretical Insights into Brain Function*, **165**, 33–56.

[32] Socher, R., Lin, C., Ng, A. Y., and Manning, C. (2011). Learning continuous phrase representations and syntactic parsing with recursive neural networks. In *ICML'2011*.

[33] Taylor, G. and Hinton, G. (2009). Factored conditional restricted Boltzmann machines for modeling motion style. In *ICML 2009*, pages 1025–1032.

[34] Taylor, G., Hinton, G. E., and Roweis, S. (2007). Modeling human motion using binary latent variables. In *NIPS'06*, pages 1345–1352. MIT Press, Cambridge, MA.

[35] Utgoff, P. E. and Stracuzzi, D. J. (2002). Many-layered learning. *Neural Computation*, **14**, 2497–2539.

[36] Weston, J., Ratle, F., and Collobert, R. (2008). Deep learning via semi-supervised embedding. In *ICML 2008*, pages 1168–1175, New York, NY, USA.

[37] Wolpert, D. H. (1996). The lack of a priori distinction between learning algorithms. *Neural Computation*, **8**(7), 1341–1390.

[38] Yao, A. (1985). Separating the polynomial-time hierarchy by oracles. In *Proceedings of the 26th Annual IEEE Symposium on Foundations of Computer Science*, pages 1–10.

